# Closed-Form Inversion of Backpropagation Networks: Theory and Optimization Issues

**Michael L. Rossen**
HNC, Inc.
5501 Oberlin Drive
San Diego, CA 92121
rossen@amos.ucsd.edu

## Abstract

We describe a closed-form technique for mapping the output of a trained backpropagation network into input activity space. The mapping is an *inverse mapping* in the sense that, when the image of the mapping in input activity space is propagated forward through the normal network dynamics, it reproduces the output used to generate that image. When more than one such inverse mappings exist, our inverse mapping is special in that it has no projection onto the nullspace of the activation flow operator for the entire network. An important by-product of our calculation, when more than one inverse mappings exist, is an orthogonal basis set of a significant portion of the activation flow operator nullspace. This basis set can be used to obtain an alternate inverse mapping that is optimized for a particular real-world application.

## 1 Overview

This paper describes a closed-form technique for mapping a particular output of a trained backpropagation network into input activity space. The mapping produced by our technique is an *inverse mapping* in the sense that, when the image in input space of the mapping of an output activity is propagated forward through the normal network dynamics, it reproduces the output used to generate it.[1] When multiple inverse mappings exist, our inverse mapping is unique in that it has no

projection onto the nullspace of the activation flow operator for the entire network. An important by-product of our calculation is an orthogonal basis set of a significant portion of this nullspace. Any vector within this nullspace can be added to the image from the inverse mapping, producing a new point in input space that is still an inverse mapping image in the above sense. Using this nullspace, the inverse mapping can be optimized for a particular application by minimizing a cost function over the input elements, relevant to that application, to obtain the vector from the nullspace to add to the original inverse mapping image. For this reason and because of the closed-form we obtain for calculation of the network inverse mapping, our method compares favorably to previously proposed iterative methods of network inversion [Widrow & Stearns, 1985, Linden & Kinderman, 1989]. We now briefly summarize our method of closed-form inversion of a backpropagation network.

## 2 The Inverse Mapping Operator

To outline the calculation of our inverse mapping operator, we start by considering a trained feed-forward backpropagation network with one hidden layer and bipolar sigmoidal activation functions. We calculate this inverse as a sequence of the inverses of the sub-operations constituting the dynamics of activation flow. If we use the 'I, II, O' as subscripts indicating input, hidden and output modules of model neurons, respectively, the activation flow from input through hidden module to output module is:

$$
\begin{aligned}
\mathbf{\underline{f}}_{(O)} &= \sigma \odot \mathcal{W}_{(O,H)} \odot \mathbf{\underline{f}}_{(H)} \\
&= \sigma \odot \mathcal{W}_{(O,H)} \odot \sigma \odot \mathcal{W}_{(H,I)} \odot \mathbf{\underline{f}}_{(I)} \\
&\doteq \mathcal{A} \odot \mathbf{\underline{f}}_{(I)},
\end{aligned}
\tag{1}
$$

where

$\sigma$ : bipolar sigmoid function;

$\mathcal{W}_{(dest,source)}$ : Matrix operator of connection weights, indexed

by 'source' and 'dest'(destination) modules;

$\mathbf{\underline{f}}_{(k)}$ : Vector of activities for module 'k'.

$\mathcal{A}$ is defined here as the activation flow operator for the entire network. The symbol $\odot$ separates operators sequentially applied to the argument.

Since the sub-operators constituting $\mathcal{A}$ are applied sequentially, the inverse that we calculate, $\mathcal{A}^+$, is equal to a composition of inverses of the individual sub-operators, with the order of the composition reversed from the order in activation flow. The closed-form mapping of a specified output $\mathbf{\underline{f}}_{(O)}$ to input space is then:

$$
\begin{aligned}
\mathbf{\underline{f}}_{(I)} &= \mathcal{A}^+ \odot \mathbf{\underline{f}}_{(O)} \\
&= \mathcal{W}_{(O,H)}^+ \odot \sigma^{-1} \odot \mathcal{W}_{(H,I)}^+ \odot \sigma^{-1} \odot \mathbf{\underline{f}}_{(O)},
\end{aligned}
\tag{2}
$$

where

$\sigma^{-1}$ : Inverse of the bipolar logistic sigmoid;

$\mathcal{W}_{(dest,source)}^+$ : Pseudo-inverse of $\mathcal{W}_{(dest,source)}$ .

Subject to the existence conditions discussed in section 4, $\hat{\mathbf{f}}_{(I)}$ is an inverse mapping of $\mathbf{f}_{(O)}$ in that it reproduces $\mathbf{f}_{(O)}$ when it is propagated forward through the network:

$$\mathbf{f}_{(O)} \;\; = \;\; \mathcal{A} \odot \hat{\mathbf{f}}_{(I)}. \tag{3}$$

We use singular value decomposition (SVD), a well-known matrix analysis method (e.g., [Lancaster, 1985]), to calculate a particular matrix inverse, the pseudo-inverse $\mathcal{W}^{+}_{(j,i)}$ (also known as the Moore-Penrose inverse) of each connection weight matrix block. In the case of $\mathcal{W}_{(H,I)}$, for example, SVD yields the two unitary matrices, $\mathcal{S}_{(H,I)}$ and $\mathcal{V}_{(H,I)}$, and a rectangular matrix $\mathcal{D}_{(H,I)}$, all zero except for the singular values on its diagonal, such that

$$\mathcal{W}_{(H,I)} \;\; = \;\; \mathcal{S}_{(H,I)}\mathcal{D}_{(H,I)}\mathcal{V}^{*}_{(H,I)} \tag{4}$$

$$\mathcal{W}^{+}_{(H,I)} \;\; = \;\; \mathcal{V}_{(H,I)}\mathcal{D}^{+}_{(H,I)}\mathcal{S}^{*}_{(H,I)}, \tag{5}$$

where

$\mathcal{V}^{*}_{(H,I)}, \mathcal{V}^{*}_{(H,I)}$ : Transposes of $\mathcal{S}_{(H,I)}$, $\mathcal{V}_{(H,I)}$, respectively;

$\mathcal{D}^{+}_{(H,I)}$ : Pseudo-inverse of $\mathcal{D}_{(H,I)}$, which is simply its transpose

with each non-zero singular value replaced by its inverse.

## 3    Uniqueness and Optimization Considerations

The pseudo-inverse (calculated by SVD or other methods) is one of a class of solutions to the inverse of a matrix operator that may exist, called generalized inverses. For our purposes, each of these generalized inverses, if they exist, are inverses in the useful sense that when substitued for $\mathcal{W}^{+}_{(j,i)}$ in eq. (2), the resultant $\hat{\mathbf{f}}_{(I)}$ will be and inverse mapping image as defined by eq. (3).

When a matrix operator $\mathcal{W}$ does not have a nullspace, the pseudo-inverse is the only generalized inverse that exists. If $\mathcal{W}$ does have a nullspace, the pseudo-inverse is special in that its range contains no projection onto the nullspace of $\mathcal{W}$. It follows that if either of the matrix operators $\mathcal{W}_{(H,I)}$ or $\mathcal{W}_{(O,H)}$ in eq. (1) have a nullspace, then multiple inverse mapping operators will exist. However, the inverse mapping operator $\mathcal{A}^{+}$ calculated using pseudo-inverses will be the only inverse mapping operator that has no projection in the nullspace of $\mathcal{A}$. The derivation of these properties follow in a straightforward manner from the discussion of generalized inverses in [Lancaster, 1985]. An interesting result of using SVD to obtain the pseudo-inverse is that:

**SVD provides a direct method for varying $\hat{\mathbf{f}}_{(I)}$ within the space of inverse mapping images in input space of $\mathbf{f}_{(O)}$.**

This becomes clear when we note that if $r = \rho(\mathcal{W}_{(H,I)})$ is the rank of $\mathcal{W}_{(H,I)}$, only the first $r$ singular values in $\mathcal{D}_{(H,I)}$ are non-zero. Thus, only the first $r$ columns of $\mathcal{S}_{(H,I)}$ and $\mathcal{V}_{(H,I)}$ participate in the activity flow of the network from input module to hidden module.

The columns $\{\underline{\mathbf{v}}_{(H,I)(i)}\}_{i>r}$ of $\mathcal{V}_{(H,I)}$ span the nullspace of $\mathcal{W}_{(H,I)}$. This nullspace is also the nullspace of $\mathcal{A}$, or at least a significant portion thereof.[2] If $\hat{\mathbf{f}}_{(I)}$ is an inverse mapping image of $\underline{\mathbf{f}}_{(O)}$, then the addition of any vector from the nullspace to $\hat{\mathbf{f}}_{(I)}$ would still be an inverse mapping image of $\underline{\mathbf{f}}_{(O)}$, satisfying eq. (3). If an inverse mapping image $\hat{\mathbf{f}}_{(I)}$ obtained from eq. (2) is unphysical, or somehow inappropriate for a particular application, it could possibly be optimized by combining it with a vector from the nullspace of $\mathcal{A}$.

## 4   Existence and Stability Considerations

There are still implementational issues of importance to address:

1. For a given $\underline{\mathbf{f}}_{(O)}$, can eq. (2) produce some mapping image $\hat{\mathbf{f}}_{(I)}$?

2. For a given $\underline{\mathbf{f}}_{(O)}$, will the image $\hat{\mathbf{f}}_{(I)}$ produced by eq. (2) be a true inverse mapping image; i.e., will it satisfy eq. (3)? If not, is it a best approximation in some sense?

3. How stable is an inverse mapping from $\underline{\mathbf{f}}_{(O)}$ that produces the answer 'yes' to questions 1 and 2; i.e., if $\underline{\mathbf{f}}_{(O)}$ is perturbed to produce a new output point, will this new output point satisfy questions 1 and 2?

In general, eq. (2) will produce an image for any output point generated by the forward dynamics of the network, eq. (1). If $\underline{\mathbf{f}}_{(O)}$ is chosen arbitrarily, however, then whether it is in the domain of $\mathcal{A}^+$ is purely a function of the network weights. The domain is restricted because the domain of the inverse sigmoid sub-operator is restricted to $(-1,+1)$.

Whether an image produced by eq. (2) will be an *inverse* mapping image, i.e., satisfying eq. (3), is dependent on both the network weights and the network architecture. A strong sufficient condition for guaranteeing this condition is that the network have a *convergent* architecture; that is:

- The dimension of input space is greater than or equal to the dimension of output space.

- The rank of $\mathcal{D}_{(H,I)}$ is greater than or equal to the rank of $\mathcal{D}_{(O,H)}$.

The stability of inverse mappings of a desired output away from such an actual output depends wholly on the weights of the network. The range of singular values of weight matrix block $\mathcal{W}_{(O,H)}$ can be used to address this issue. If the range is much more than one order of magnitude, then random perturbations about a given point in output space will often be outside the domain of $\mathcal{A}^+$. This is because the columns of $\mathcal{S}_{(O,H)}$ and $\mathcal{V}_{(O,H)}$ associated with small singular values during forward

activity flow are associated with proportionately large inverse singular values in the inverse mapping. Thus, if singular value $d_{O,Hi}$ is small, a random perturbation with a projection on column $\underline{s}_{(O,H)(i)}$ of $\mathcal{S}_{(O,H)}$ will cause a large magnitude swing in the inverse sub-operator $\mathcal{W}^+_{(O,H)}$, with the result possibly outside the domain of $\sigma^{-1}$.

## 5    Summary

- We have shown that a closed-form inverse mapping operator of a backpropagation network can be obtained using a composition of pseudo-inverses and inverse sigmoid operators.

- This inverse mapping operator, specified in eq. (2), operating on any point in the network's output space, will obtain an inverse image of that point that satisfies eq. (3), if such an inverse image exists.

- When many inverse images of an output point exist, an extension of the SVD analyses used to obtain the original inverse image can be used to obtain an alternate inverse image optimized to satisfy the problem constraints of a particular application.

- The existence of an inverse image of a particular output point depends on that output point and the network weights. The dependence on the network can be expressed conveniently in terms of the singular values and the singular value vectors of the network weight matrices.

- Application for these techniques include explanation of network operation and process control.

## Footnotes

[1] It is possible that no such inverse mappings exist. This point is addressed in section 4.

[2]Since its first sub-operation is linear, and the sigmoid non-linearity we employ maps zero to zero, the non-linear operator $\mathcal{A}$ can still have a nullspace. Subsequent layers of the network might add to this nullspace, however, and the added region may not be a linear subspace.

## References

[Lancaster, 1985]            Lancaster, P., & Tismenetsky, M. (1985). *The Theory of Matrices*. Orlando: Academic.

[Linden & Kinderman, 1989]   Linden, A., & Kinderman, J. (1989). Inversion of multilayer nets. *Proceedings of the Third Annual International Joint Conference on Neural Networks, Vol II*, 425-430.

[Widrow & Stearns, 1985]     Widrow, B., & Stearns, S.D. (1985). *Adpative Signal Processing*. Englewood Cliffs: Prentice-Hall.


